# Propagation Filters in PDS Networks for Sequencing and Ambiguity Resolution

**Ronald A. Sumida**
**Michael G. Dyer**
Artificial Intelligence Laboratory
Computer Science Department
University of California
Los Angeles, CA, 90024
sumida@cs.ucla.edu

## Abstract

We present a Parallel Distributed Semantic (PDS) Network architecture that addresses the problems of sequencing and ambiguity resolution in natural language understanding. A PDS Network stores phrases and their meanings using multiple PDP networks, structured in the form of a semantic net. A mechanism called Propagation Filters is employed: (1) to control communication between networks, (2) to properly sequence the components of a phrase, and (3) to resolve ambiguities. Simulation results indicate that PDS Networks and Propagation Filters can successfully represent high-level knowledge, can be trained relatively quickly, and provide for parallel inferencing at the knowledge level.

## 1  INTRODUCTION

Backpropagation has shown considerable potential for addressing problems in natural language processing (NLP). However, the traditional PDP [Rumelhart and McClelland, 1986] approach of using one (or a small number) of backprop networks for NLP has been plagued by a number of problems: (1) it has been largely unsuccessful at representing high-level knowledge, (2) the networks are slow to train, and (3) they are sequential at the knowledge level. A solution to these problems is to represent high-level knowledge structures over a large number of smaller PDP net-

works. Reducing the size of each network allows for much faster training, and since the different networks can operate in parallel, more than one knowledge structure can be stored or accessed at a time.

In using multiple networks, however, a number of important issues must be addressed: how the individual networks communicate with one another, how patterns are routed from one network to another, and how sequencing is accomplished as patterns are propagated. In previous papers [Sumida and Dyer, 1989] [Sumida, 1991], we have demonstrated how to represent high-level semantic knowledge and generate dynamic inferences using Parallel Distributed Semantic (PDS) Networks, which structure multiple PDP networks in the form of a semantic network. This paper discusses how Propagation Filters address communication and sequencing issues in using multiple PDP networks for NLP.

## 2    PROPAGATION FILTERS

Propagation Filters are inspired by the idea of skeleton filters, proposed by [Sejnowski, 1981, Hinton, 1981]. They are composed of: (1) sets of filter ensembles that gate the connection from a source to a destination and (2) a selector ensemble that decides which filter group to enable. Each filter group is sensitive to a particular pattern over the selector. When the particular pattern occurs, the source pattern is propagated to its destination. Figure 1 is an example of a propagation filter where the "01" pattern over units 2 and 3 of the selector opens up filter group1, thus permitting the pattern to be copied from source1 to destination1. The units of filter group2 do not respond to the "01" pattern and remain well below thresold, so the activation pattern over the source2 ensemble is not propagated.

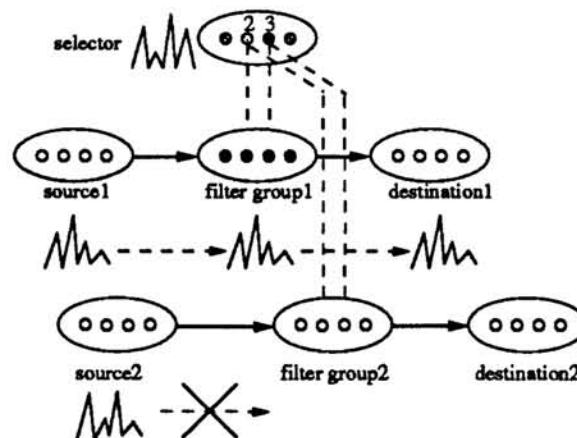

Figure 1: A Propagation Filter architecture. The small circles indicate PDP units within an ensemble (oval), the black arrows represent full connectivity between two ensembles, and the dotted lines connecting units 2 and 3 of the selector to each filter group oval indicate total connectivity from selector units to filter units. The jagged lines are suggestive of temporary patterns of activation over an ensemble.

The units in a filter group receive input from units in the selector. The weights on these input connections are set so that when a specific pattern occurs over the

selector, every unit in the filter group is driven above threshold. The filter units also receive input from the source units and provide output to the destination units. The weights on both these i/o connections can be set so that the filter merely *copies* the pattern from the source to the destination when its units exceed threshold (as in Figure 1). Alternatively, these weights can be set (e.g. using backpropagation) so that the filter *transforms* the source pattern to a desired destination pattern.

## 3    PDS NETWORKS

PDS Networks store syntactic and semantic information over multiple PDP networks, with each network representing a class of concepts and with related networks connected in the general manner of a semantic net. For example, Figure 2 shows a network for encoding a basic sentence consisting of a subject, verb and direct object. The network is connected to other PDP networks, such as HUMAN, VERB and ANIMAL, that store information about the content of the subject role (s-content), the filler for the verb role, and the content of the direct-object role (do-content). Each network functions as a type of *encoder net*, where: (1) the input and output layers have the same number of units and are presented with exactly the same pattern, (2) the weights of the network are modified so that the input pattern will recreate itself as output, and (3) the resulting hidden unit pattern represents a *reduced description* of the input. In the networks that we use, a single set of units is used for both the input and output layers. The net can thus be viewed as an encoder with the output layer folded back onto the input layer and with two sets of connections: one from the single input/output layer to the hidden layer, and one from the hidden layer back to the i/o layer. In Figure 2 for example, the subject-content, verb, and direct-object-content role-groups collectively represent the input/output layer, and the BASIC-S ensemble represents the hidden layer.

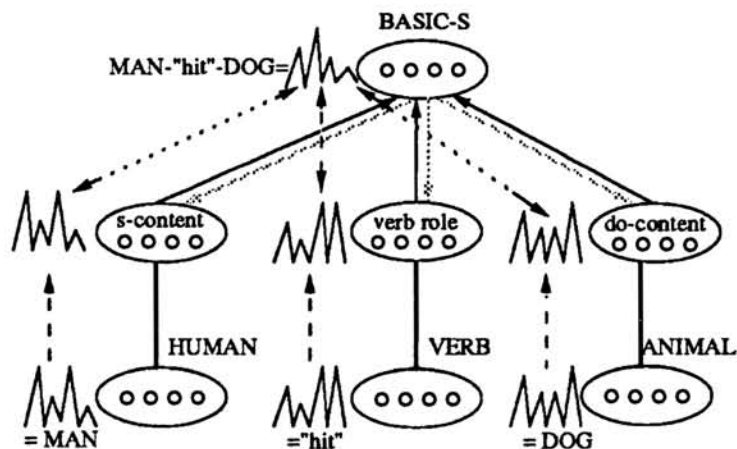

Figure 2: The network that stores information about a basic sentence. The black arrows represent links from the input layer to the hidden layer and the grey arrows indicate links from the hidden layer to the output layer. The thick lines represent links between networks that propagate a pattern without changing it.

A network stores information by learning to encode the items in its training set.

For each item, the patterns that represent its features are presented to the input role groups, and the weights are modified so that the patterns recreate themselves as output. For example, in Figure 2, the MAN-"hit"-DOG pattern is presented to the BASIC-S network by propagating the MAN pattern from the HUMAN network to the s-content role, the "hit" pattern from the VERB network to the verb-content role, and the DOG pattern from the ANIMAL network to the do-content role. The BASIC-S network is then trained on this pattern by modifying the weights between the input/output role groups and the BASIC-S hidden units so that the MAN-"hit"-DOG pattern recreates itself as output. The network automatically generalizes by having the hidden units become sensitive to common features of the training patterns. When the network is tested on a new concept (i.e., one that is not in the training set), the pattern over the hidden units reflects its similarity to the items seen during training.

## 3.1   SEQUENCING PHRASES

To illustrate how Propagation Filters sequence the components of a phrase, consider the following sentence, whose constituents occur in the standard subject-verb-object order: *S1. The man hit the dog.* We would like to recognize that the BASIC-S network of Figure 2 is applicable to the input by binding the roles of the network to the correct components. In order to generate the proper role bindings, the system must: (1) recognize the components of the sentence in the correct order (e.g. "the man" should be recognized as the subject, "hit" as the verb, and "the dog" as the direct object), and (2) associate each phrase of the input with its meaning (e.g. reading the phrase "the man" should cause the pattern for the concept MAN to appear over the HUMAN units). Figure 3 illustrates how Propagation Filters properly sequence the components of the sentence.

First, the phrase "the man" is read by placing the pattern for "the" over the determiner network (Step 1) and the pattern for "man" over the noun network (Step 2). The "the" pattern is then propagated to the np-determiner input role units of the NP network (Step 3) and the "man" pattern to the np-noun role input units (Step 4). The pattern that results over the hidden NP units is then used to represent the entire phrase "the man" (Step 5). The filters connecting the NP units with the subject and direct object roles are not enabled, so the pattern is not yet bound to any role. Next, the word "hit" is read and a pattern for it is generated over the VERB units (Step 6). The BASIC-S network is now applicable to the input (for simplicity of exposition, we ignore passive constructions here). Since there are no restrictions (i.e., no filter) on the connection between the VERB units and the verb role of BASIC-S, the "hit" pattern is bound to the verb role (Step 7). The verb role units act as the selector of the Propagation Filter that connects the NP units to the subject units. The filter is constructed so that whenever any of the verb role units receive non-zero input (i.e., whenever the role is bound) it opens up the filter group connecting NP with the subject role (Step 8). Thus, the pattern for "the man" is copied from NP to the subject (Step 9) and deleted from the NP units. Similarly, the subject units act as the selector of a filter that connects NP with the direct object. Since the subject was just bound, the connection from the NP to direct object is enabled (Step 10). At this point, the system has generated the *expectation* that a NP will occur next. The phrase "the dog" is now read and

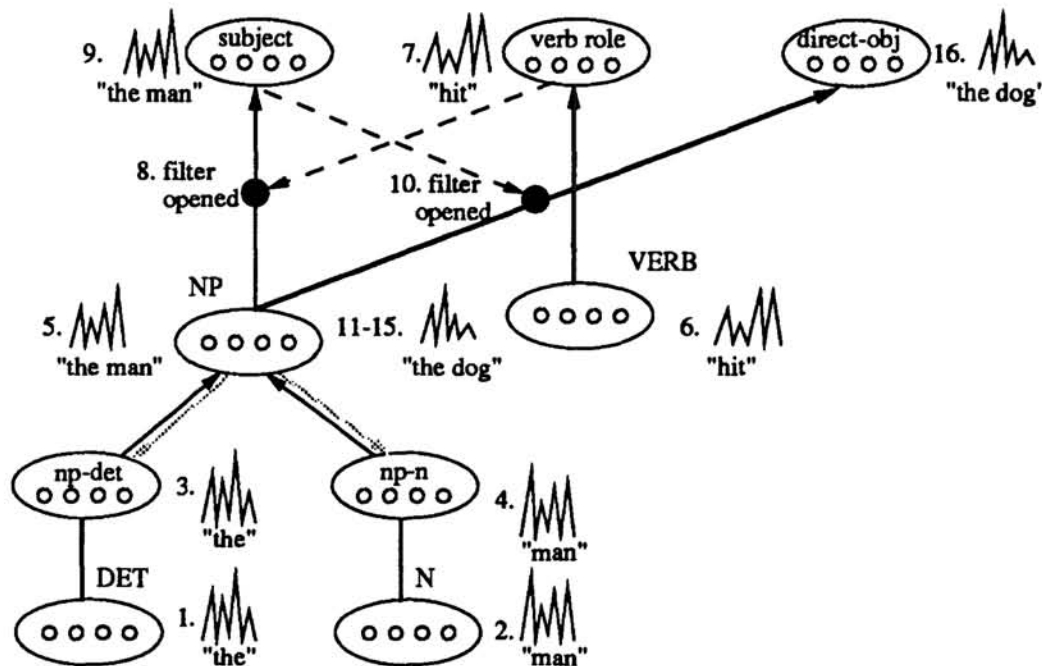

Figure 3: The figure shows how Propagation Filters sequence the components of the sentence "The man hit the dog". The numbers indicate the order of events. The dotted arrows indicate Propagation Filter connections from a selector to an open filter group (indicated by a black circle) and the dark arrows represent the connections from a source to a destination.

its pattern is generated over the NP units (Steps 11-15). Finally, the pattern for "the dog" is copied across the open connection from NP to direct-object (Step 16).

## 3.2   ASSOCIATING PHRASES WITH MEANINGS

The next task is to associate lexical patterns with their corresponding semantic patterns and bind semantic patterns to the appropriate roles in the BASIC-S network. Figure 4 indicates how Propagation Filters: (1) transform the phrase "the man" into its meaning (i.e., MAN), and (2) bind MAN to the s-content role of BASIC-S.

Reading the word "man", by placing the "man" pattern into the noun units (Step 2), opens the filter connecting N to HUMAN (Step 5), while leaving the filters connecting N to other networks (e.g. ANIMAL) closed. The opened filter transforms the lexical pattern "man" over N into the semantic pattern MAN over HUMAN (Step 7). Binding "the man" to subject (Step 8) by the procedure shown in the Figure 3 opens the filter connecting HUMAN to the s-content role of BASIC-S (Step 9). The s-content role is then bound to MAN (Step 10).

The do-content role is bound by a procedure similar to that shown in Figure 4. When "dog" is read, the filter connecting N with ANIMAL is opened while filters to other networks (e.g. HUMAN) remain closed. The "dog" pattern is then transformed into the semantic pattern DOG over the ANIMAL units. When "the dog"

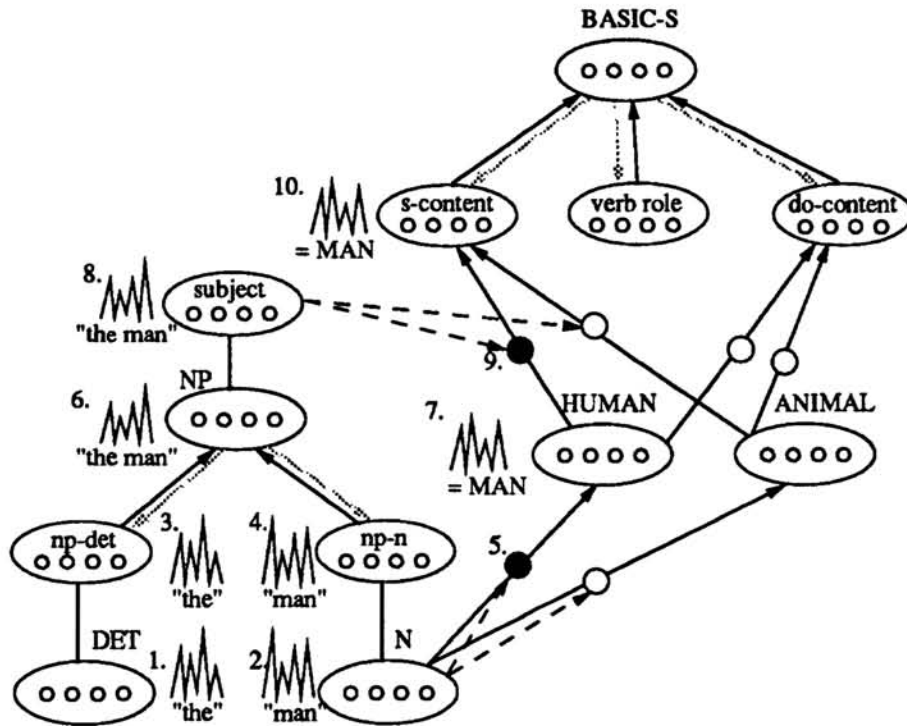

Figure 4: The figure illustrates how the concept MAN is bound to the s-content role of BASIC-S, given the phrase "the man" as input. Black (white) circles indicate open (closed) filters.

is bound to direct-object as in Figure 3, the filter from ANIMAL to do-content is opened, and DOG is propagated from ANIMAL to the do-content role of BASIC-S.

## 3.3   AMBIGUITY RESOLUTION AND INFERENCING

There are two forms that inference and ambiguity resolution can take: (1) routing patterns (e.g. propagation of role bindings) to the appropriate subnets and (2) pattern reconstruction from items seen during training.

(1) *Pattern Routing*: Propagation Filters help resolve ambiguities by having the selector only open connections to the network containing the correct interpretation. As an example, consider the following sentence: *S2. The singer hit the note.* Both S2 and S1 (Sec. 3.1) have the same syntactic structure and are therefore represented over the BASIC-S ensemble of Figure 2. However, the meaning of the word "hit" in S1 refers to physically striking an object while in S2 it refers to singing a musical note. The pattern over the BASIC-S units that represents S1 differs significantly from the pattern that represents S2, due to the differences in the s-content and do-content roles. A Propagation Filter with the BASIC-S units as its selector uses the differences in the two patterns to determine whether to open connections to the HIT network or to the PERFORM-MUSIC network (Figure 5).

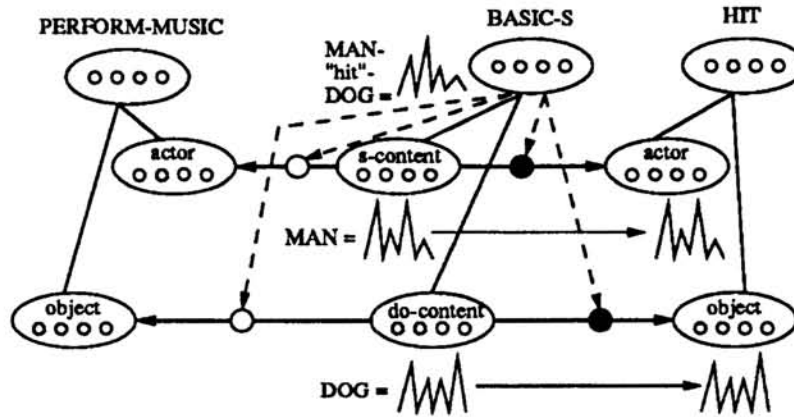

Figure 5: The pattern over BASIC-S acts as a selector that determines whether to open the connections to HIT or to PERFORM-MUSIC. Since the input here is MAN-"hit"-DOG, the filters to HIT are opened while the filters to PERFORM-MUSIC remain closed. The black and grey arrows indicating connections between the input/output and hidden layers have been replaced by a single thin line.

During training, the BASIC-S network was presented with sentences of the general form <MUSIC-PERFORMER "hit" MUSICAL-NOTE> and <ANIMATE "hit" OBJECT>. The BASIC-S hidden units generalize from the training sentences by developing a distinct pattern for each of the two types of "hit" sentences. The Propagation Filter is then constructed so that the hidden unit pattern for <MUSIC-PERFORMER "hit" MUSICAL-NOTE> opens up connections to PERFORM-MUSIC, while the pattern for <ANIMATE "hit" OBJECT> opens up connections to HIT. Thus, when S1 is presented, the BASIC-S hidden units develop the pattern classifying it as <ANIMATE "hit" OBJECT>, which enables connections to HIT. For example, Figure 5 shows how the MAN pattern is routed from the s-content role of BASIC-S to the actor role of HIT and the DOG pattern is routed from the do-content role of BASIC-S to the object role of HIT. If S2 is presented instead, the hidden units will classify it as <MUSIC-PERFORMER "hit" MUSICAL-NOTE> and open the connections to PERFORM-MUSIC.

The technique of using propagation filters to control pattern routing can also be applied to generate inferences. Consider the sentence, "Douglas hit Tyson". Since both are boxers, it is plausible they are involved in a competitive activity. In S1, however, punishing the dog is a more plausible motivation for HIT. The proper inference is generated in each case by training the HIT network (Figure 5) on a number of instances of boxers hitting one another and of people hitting dogs. The network learns two distinct sets of hidden unit patterns: <BOXER-HIT-BOXER> and <HUMAN-HIT-DOG>. A Propagation Filter, (like that shown in Figure 5) with the HIT units as its selector, uses the differences in the two classes of patterns to route to either the network that stores competitive activities or to the network that stores punishment acts.

(2) *Pattern Reconstruction*: The system also resolves ambiguities by reconstructing patterns that were seen during training. For example, the word "note" in sentence

S2 is ambiguous and could refer to a message, as in "The singer left the note". Thus, when the word "note" is read in S2, the do-content role of BASIC-S can be bound to MESSAGE or to MUSICAL-NOTE. To resolve the ambiguity, the BASIC-S network uses the information that SINGER is bound to the s-content role and "hit" to the verb role to: (1) reconstruct the <MUSIC-PERFORMER "hit" MUSICAL-NOTE> pattern that it learned during training and (2) predict that the do-content will be MUSICAL-NOTE. Since the prediction is consistent with one of the possible meanings for the do-content role, the ambiguity is resolved. Similarly, if the input had been "The singer left the note", BASIC-S would use the binding of a human to the s-content role and the binding of "left" to the verb role to reconstruct the pattern <HUMAN "left" MESSAGE> and thus resolve the ambiguity.

## 4    CURRENT STATUS AND CONCLUSIONS

PDS Networks and Propagation Filters are implemented in DCAIN, a natural language understanding system that: (1) takes each word of the input sequentially, (2) binds the roles of the corresponding syntactic and semantic structures in the proper order, and (3) resolves ambiguities. In our simulations with DCAIN, we successfully represented high-level knowledge by structuring individual PDP networks in the form of a semantic net. Because the system's knowledge is spread over multiple subnetworks, each one is relatively small and can therefore be trained quickly. Since the subnetworks can operate in parallel, DCAIN is able to store and retrieve more than one knowledge structure simultaneously, thus achieving knowlege-level parallelism. Because PDP ensembles (versus single localist units) are used, the generalization, noise and fault-tolerance properties of the PDP approach are retained. At the same time, Propagation Filters provide control over the way patterns are routed (and transformed) between subnetworks. The PDS architecture, with its Propagation Filters, thus provides significant advantages over traditional PDP models for natural language understanding.

**References**

[Hinton, 1981] G. E. Hinton. Implementing Semantic Networks in Parallel Hardware. In *Parallel Models of Associative Memory*, Lawrence Erlbaum, Hillsdale, NJ, 1981.

[Rumelhart and McClelland, 1986] D. E. Rumelhart and J. L. McClelland. *Parallel Distributed Processing*, Volume 1. MIT Press, Cambridge, Massachusetts, 1986.

[Sejnowski, 1981] T. J. Sejnowski. Skeleton Filters in the Brain. In *Parallel Models of Associative Memory*, Lawrence Erlbaum, Hillsdale, NJ, 1981.

[Sumida and Dyer, 1989] R. A. Sumida and M. G. Dyer. Storing and Generalizing Multiple Instances while Maintaining Knowledge-Level Parallelism. In *Proceedings of the Eleventh International Joint Conference on Artificial Intelligence*, Detroit, MI, 1989.

[Sumida, 1991] R. A. Sumida. Dynamic Inferencing in Parallel Distributed Semantic Networks. In *Proceedings of the Thirteenth Annual Conference of the Cognitive Science Society*, Chicago, IL, 1991.
